# Transformation Invariant Autoassociation with Application to Handwritten Character Recognition

**Holger Schwenk**          **Maurice Milgram**

**PARC**
Université Pierre et Marie Curie
tour 66-56, boite 164
4, place Jussieu, 75252 Paris cedex 05, France.
e-mail: schwenk@robo.jussieu.fr

## Abstract

When training neural networks by the classical backpropagation algorithm the whole problem to learn must be expressed by a set of inputs and desired outputs. However, we often have high-level knowledge about the learning problem. In optical character recognition (OCR), for instance, we know that the classification should be invariant under a set of transformations like rotation or translation. We propose a new modular classification system based on several autoassociative multilayer perceptrons which allows the efficient incorporation of such knowledge. Results are reported on the NIST database of upper case handwritten letters and compared to other approaches to the invariance problem.

## 1   INCORPORATION OF EXPLICIT KNOWLEDGE

The aim of supervised learning is to learn a mapping between the input and the output space from a set of example pairs (input, desired output). The classical implementation in the domain of neural networks is the backpropagation algorithm. If this learning set is sufficiently representative of the underlying data distributions, one hopes that after learning, the system is able to generalize correctly to other inputs of the same distribution.

It would be better to have more powerful techniques to incorporate knowledge into the learning process than the choice of a set of examples. The use of additional knowledge is often limited to the feature extraction module. Besides simple operations like (size) normalization, we can find more sophisticated approaches like zernike moments in the domain of optical character recognition (OCR). In this paper we will not investigate this possibility, all discussed classifiers work directly on almost non preprocessed data (pixels).

In the context of OCR interest focuses on invariance of the classifier under a number of given transformations (translation, rotation, ...) of the data to classify. In general a neural network could extract those properties of a large enough learning set, but it is very hard to learn and will probably take a lot of time. In the last years two main approaches for this invariance problem have been proposed: *tangent-prop* and *tangent-distance*. An indirect incorporation can be achieved by *boosting* (Drucker, Schapire and Simard, 1993).

In this paper we briefly discuss these approaches and will present a new classification system which allows the efficient incorporation of transformation invariances.

## 1.1 TANGENT PROPAGATION

The principle of tangent-prop is to specify besides desired outputs also desired changes $j^\mu$ of the output vector when transforming the net input $x$ by the transformations $t_\mu$ (Simard, Victorri, LeCun and Denker, 1992).

For this, let us define a transformation of pattern $p$ as $t(p, \alpha) : P \to P$ where $P$ is the space of all patterns and $\alpha$ a parameter. Such transformations are in general highly nonlinear operations in the pixel space $P$ and their analytical expressions are seldom known. It is therefore favorable to use a first order approximation:

$$t(p, \alpha) \approx p + \alpha\, t_p \quad \text{with} \quad t_p = \left. \frac{\partial t(p, \alpha)}{\partial \alpha} \right|_{\alpha=0} \tag{1}$$

$t_p$ is called the *tangent vector*. This definition can be generalized to $c$ transformations:

$$t(p, \vec{\alpha}) \approx p + \alpha_1\, t_{p1} + \ldots + \alpha_c\, t_{pc} = p + T_p \vec{\alpha} \tag{2}$$

where $T_p$ is a $n \times c$ matrix, each column corresponding to a tangent vector.

Let us define $R(x)$ the function calculated by the network. The desired behavior of the net outputs can be obtained by adding a regularization term $E_r$ to the objective function:

$$E_r = \sum_{\mu=1}^{c} \left\| j^\mu - \left. \frac{\partial R(t^\mu(x, \vec{\alpha}))}{\partial \vec{\alpha}} \right|_{\vec{\alpha}=0} \right\|^2 \approx \sum_{\mu=1}^{c} \left\| j^\mu - \frac{\partial R(x)}{\partial x} t_x^\mu \right\|^2 \tag{3}$$

$t_x^\mu$ is the tangent vector for transformation $t^\mu$ of the input vector $x$ and $\partial R(x)/\partial x$ is the gradient of the network with respect to the inputs. Transformation invariance of the outputs is obtained by setting $j^\mu = 0$, so we want that $\partial R(x)/\partial x$ is orthogonal to $t_x^\mu$.

Tangent-prop improved the learning time and the generalization on small databases, but its applicability to highly constraint networks (many shared weights) trained on large databases remains unknown.

## 1.2 TANGENT DISTANCE

Another class of classifiers are memory based learning methods which rely on distance metrics. The incorporation of knowledge in such classifiers can be done by a distance

measure which is (locally) invariant under a set of specified transformations.

(Simard, LeCun and Denker, 1993) define *tangent distance* as the minimal distance between the two hyperplanes spanned up by the tangent vectors $T_p$ in point $p$ and $T_q$ in point $q$:

$$D_{pq}(p,q) = \min_{\vec{\alpha},\vec{\beta}} \left(p + T_p\vec{\alpha} - q - T_q\vec{\beta}\right)^2 = \left(p + T_p\vec{\alpha}^* - q - T_q\vec{\beta}^*\right)^2 \qquad (4)$$

The optimality condition is that the partial derivatives $\partial D_{pq}/\partial\vec{\alpha}^*$ and $\partial D_{pq}/\partial\vec{\beta}^*$ should be zero. The values $\vec{\alpha}^*$ and $\vec{\beta}^*$ minimizing (4) can be computed by solving these two linear systems numerically.

(Simard, LeCun and Denker, 1993) obtained very good results on handwritten digits and letters using tangent distance with a 1-nearest-neighbor classifier (1-nn) . A big problem of every nn-classifier, however, is that it uses no compilation of the data and it needs therefore numerous reference vectors resulting in long classification time and high memory usage.

Like reported in (Simard, 1994) and (Sperdutti and Stork, 1995) important improvements are possible, but often a trade-off between speed and memory usage must be made.

## 2 ARCHITECTURE OF THE CLASSIFIER

The main idea of our approach is to use an *autoassociative multilayer perceptron* with a low dimensional hidden layer for each class to recognize. These networks, called *diabolo network* in the following, are trained only with examples of the corresponding class. This can be seen as supervised learning without counter-examples. Each network learns a hidden layer representation which preserves optimally the information of the examples of *one* class. These learned networks can now be used like discriminant functions: the reconstruction error is in general much lower for examples of the learned class than for the other ones.

In order to build a classifier we use a decision module which interprets the distances between the reconstructed output vectors and the presented example. In our studies we have used until now a simple minimum operator which associates the class of the net with the smallest distance (Fig. 1).

The figure illustrates also typical classification behavior, here when presenting a "D" out of the test set. One can see clearly that the distance of the network "D" is much lower than for the two other ones. The character is therefore correctly classified. It is also interesting to analyze the outputs of the two networks with the next nearest distances: the network "O" tries to output a more round character and the network "B" wants to add a horizontal bar in the middle.

The basic classification architecture can be adapted in two ways to the problem to be solved. One on hand we can imagine different architectures for each diabolo network, e.g. several encoding/decoding layers which allow nonlinear dimension reduction. It is even possible to use shared weights realizing local feature detectors (see (Schwenk and Milgram, 1994) for more details).

One the other hand we can change the underlying distance measure, as long as the derivatives with respect to the weights can be calculated. This offers a powerful and efficient mechanism to introduce explicit knowledge into the learning algorithm of a neural network. In the discussed case, the recognition of characters represented as pixel images, we can use a

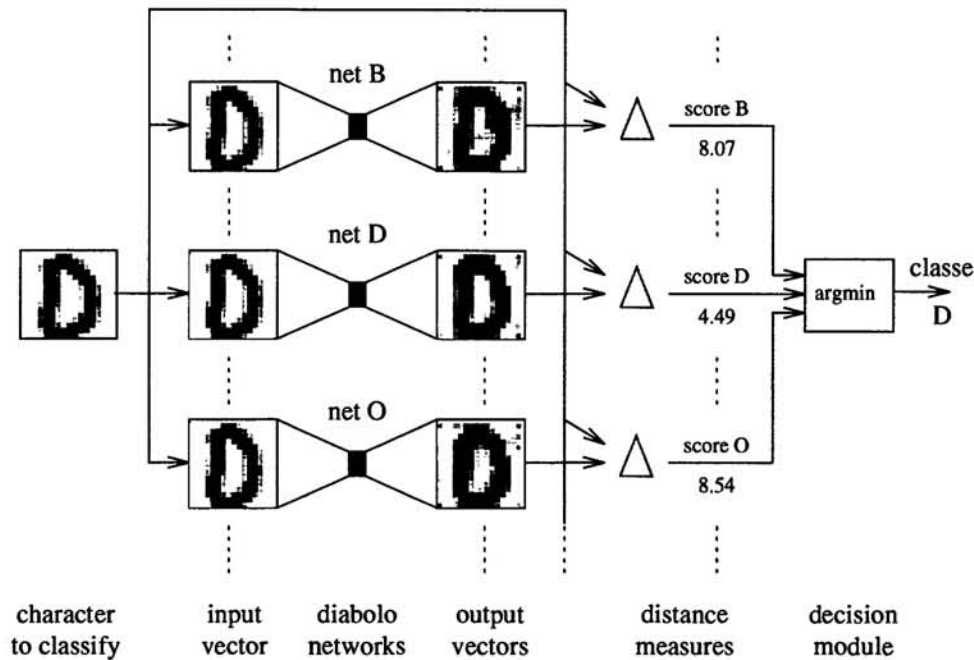

Figure 1: Basic Architecture of a Diabolo Classifier

transformation invariant distance measure between the net output $o$ and the desired output $d$ (that is of course identical with the net input). The networks do not need to learn each example separately any more, but they can use the set of specified transformations in order to find a common *non linear* model of each class.

The advantage of this approach, besides a better expected generalization behavior of course, is a very low additional complexity. In comparison to the original k-nn approach, and supposedly any possible optimization, we need to calculate only one distance measure for each class to recognize, regardless of the number of learning data.

We used two different versions of tangent distance with increasing complexity:

1. **single sided tangent distance:**

$$D_d(d, o) = \min_{\vec{\alpha}} \frac{1}{2} \left( d + T_d\vec{\alpha} - o \right)^2 = \frac{1}{2} \left( d + T_d\vec{\alpha}^* - o \right)^2 \quad (5)$$

This is the minimal distance between the hyperplane spanned up by the tangent vectors $T_d$ in input vector $d$ and the *untransformed* output vector $o$.

2. **double sided tangent distance:**

$$D_{do}(d, o) = \min_{\vec{\alpha}, \vec{\beta}} \frac{1}{2} \left( d + T_d\vec{\alpha} - o * g - T_o\vec{\beta} \right)^2 \quad (6)$$

The convolution of the net output with a Gaussian $g$ is necessary for the computation of the tangent vectors $T_o$ (the net input $d$ is convolved during preprocessing).

Figure 2 shows a graphical comparison of Euclidean distance with the two tangent distances.

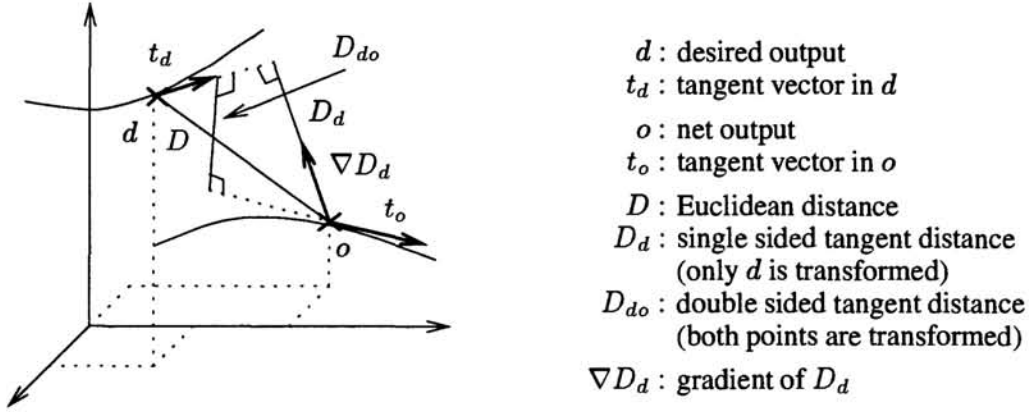

Figure 2: Comparison of Euclidean Distance with the Different Tangent Distances

The major advantage of the single sided version is that we can now calculate easily the optimal multipliers $\vec{\alpha}^*$ and therefore the whole distance (the double sided version demands expensive matrix multiplications and the numerical solution of a system of linear equations). The optimality condition $\partial D_d(d, o)/\partial \vec{\alpha}^* \stackrel{!}{=} 0^T$ gives:

$$\vec{\alpha}^* = T_{dd}^{-1} T_d^T (o - d) \tag{7}$$

The tangent vectors $T_d$ and the matrix $T_{dd}^{-1} = (T_d^T T_d)^{-1}$ can be precomputed and stored in memory. Note that it is the same for all diabolo networks, regardless of their class.

## 2.1 LEARNING ALGORITHM

When using a tangent distance with an autoencoder we must calculate its derivatives with respect to the weights, i.e. after application of the chain rule with respect to the output vector $o$. In the case of the single sided tangent distance we get:

$$-\frac{\partial D_d}{\partial o} = \left( d + T_d \vec{\alpha}^* - o \right)^T \left( T_d \frac{\partial \vec{\alpha}^*}{\partial o} - I \right) = (d + T_d \vec{\alpha}^* - o)^T \tag{8}$$

The resulting learning algorithm is therefore barely more complicated than with standard Euclidean error. Furthermore it has a pleasant graphical interpretation: the net output doesn't approach directly the desired output any more, but it takes the shortest way towards the tangent hyperplane (see also fig. 2).

The derivation of the double sided tangent distance with respect to the net output is more complicated. In particular we must derivate the convolution of the net output with a Gaussian as well as the tangent vectors $T_o$. These equations will be published elsewhere.

Training of the whole system is stopped when the error on the cross validation set reaches a minimum. Using stochastic gradient descent convergence is typically achieved after some ten iterations.

# 3   APPLICATION TO CHARACTER RECOGNITION

In 1992 the National Institute of Standards and Technology provided a Database of hand-written digits and letters, known under the name NIST Special-Database 3. This database contains about 45 000 upper case segmented characters which we have divided into learning and cross-validation set (60%) and test set (40%) respectively.

We only applied a very simple preprocessing: the binary characters were centered and size-normalized (the aspect-ratio was kept). The net input is $16 \times 16$ pixels with real-values.

## 3.1   EXPERIMENTAL RESULTS

All the following results were obtained by fully connected diabolo networks with one low dimensional hidden layer, and a set of eight transformations (x- and y-translation, rotation, scaling, axis-deformation, diagonal-deformation, x- and y-thickness). Figure 3 illustrates how the networks use the transformations.

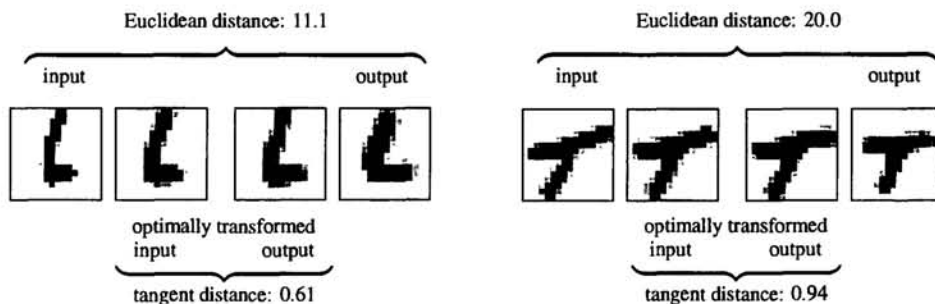

Figure 3: Reconstruction Examples (test set). The left side of each screen dump depicts the input character and the right side the one reconstructed by the network. In the middle, finally, one can see the optimally transformed patterns as calculated when evaluating the double sided tangent distance, i.e. transformed by $\vec{\alpha}^*$ and $\vec{\beta}^*$ respectively.

Although the "L" in the first example has an unusual short horizontal line, the network reconstructs a normally sized character. It is clearly visible how the input transformation lengthens and the output transformation shortens this line in order to get a small tangent distance. The right side shows a very difficult classification problem: a heavily deformed "T". Nevertheless we get a small tangent distance, so that the character is correctly classified. In summary we note a big difference between the Euclidean and the tangent distances, this is a good indicator that the networks really use the transformations.

The performances on the whole test set of about 18 000 characters are summarized in figure 4. For comparison we give also the results of a one nearest neighbor classifier on the same test set. The incorporation of knowledge improved in both cases dramatically the performances. The diabolo classifier, for instance, achieves an error rate of 4.7 % with simple Euclidean distance which goes down to 3.7 % with the single sided and to only 2.6 % with the double sided tangent distance. In order to get the same results with the 1-nn approach, the whole set of 27 000 reference vectors had to be used. It's worth to note the results with less references: when using only 18 000 reference vectors the error rates increased to 3.7% for the single sided and to 2.8% for the double sided version respectively.

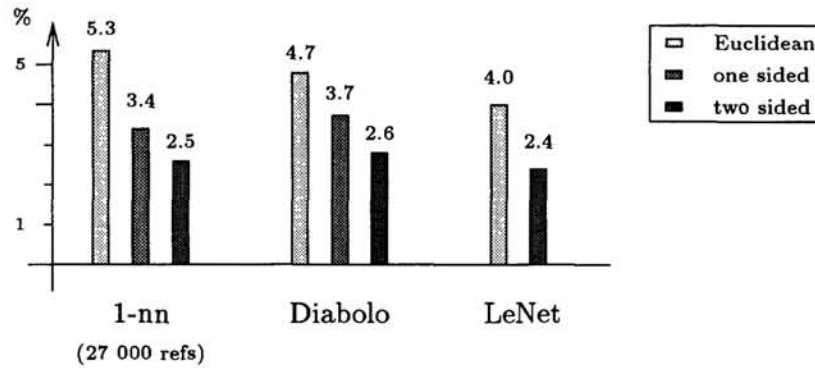

Figure 4: Raw Error Rate with NIST Upper Case Letters (test set)

In practical applications we are not only interested in low error rates, but we need also low computational costs. An important factor is the recognition speed. The overall processing time of a diabolo classifier using the full tangent distance corresponds to the calculation of about 7 000 Euclidean and to less than 50 tangent distances. This should be less than for any algorithm of the k-nn family. If we assume the precalculation of all the tangent vectors and other expensive matrix multiplications, we could evaluate about 80 tangent distances, but the price would be exploding memory requirements. A diabolo classifier, on the other hand, needs only few memory: the storage of the weights corresponds to about 60 reference vectors per class. On a HP 715/50 workstation we obtained a recognition rate of 7.5 ch/s with the single sided and of more than 2.5 ch/s with the double sided tangent distance. We have also a method to combine both by rejection, resulting in up to 4 ch/s at the same low error rates (corresponds to the calculation of 32 double sided tangent distances).

The table contains also the results of a large multilayer perceptron with extensive use of shared weights, known as LeNet. (Drucker, Schapire and Simard, 1993) give an error rate of 4.0% when used alone and of 2.4% for an ensemble of three such networks trained by boosting. The networks were trained on a basic set of 10 000 examples, the cross validation and test set consisted of 2 000 and 3 000 examples respectively (Drucker, personal communication). Due to the different number of examples, the results are perhaps not exactly comparable, but we can deduce nevertheless that the state of the art on this database seems to be around 2.5 %.

## 4  DISCUSSION

We have proposed a new classification architecture that allows the efficient incorporation of knowledge into the learning algorithm. The system is easy to train and only one structural parameter must be chosen by the supervisor: the size of the hidden layer. It achieved state of the art recognition rates on the NIST database of handwritten upper case letters at a very low computational complexity.

Furthermore we can say that a hardware implementation seems to be promising. Fully connected networks with only two layers are easy to put into standardized hardware chips. We could even propagate all diabolo networks in parallel. Speedups of several orders of magnitude should therefore be possible.

On this year NIPS conference several authors presented related approaches. A comparable classification architecture was proposed by (Hinton, Revow and Dayan, 1995). Instead of one non-linear global model per class, several local linear models were used by performing separately principal component analysis (PCA) on subsets of each class. Since diabolo networks with one hidden layer and linear activation functions perform PCA, this architecture can be interpreted as an hierarchical diabolo classifier with linear nets and Euclidean distance. Such an hierarchisation could also be done with our classifier, i.e. with tangent distance and sigmoidal units, and might improve the results even further.

(Hastie, Simard and Säckinger, 1995) developed an iterative algorithm that learns optimal reference vectors in the sense of tangent distance. An extension allows also to learn typical invariant transformations, i.e. tangent vectors, of each class. These two algorithms allowed to reduce drastically the number of reference vectors, but the results of the original approach couldn't be achieved no longer.

## Acknowledgements

The first author is supported by the German Academic Exchange Service under grant HSP II 516.006.512.3. The simulations were performed with the Aspirin/ MIGRAINES neural network simulator developed by the MITRE Corporation.

## References

H. Drucker, R. Schapire, and P. Simard (1993), "Boosting performance in neural networks," *Int. Journal of Pattern Recognition and Artificial Intelligence*, vol. 7, no. 4, pp. 705–719.

T. Hastie, P. Simard and E. Säckinger (1995), "Learning prototype models for tangent distance," in *NIPS 7* (G. Tesauro, D. Touretzky, and T. Leen, eds.), Morgan Kaufmann.

G. Hinton, M. Revow, and P. Dayan (1995), "Recognizing handwritten digits using mixtures of linear models," in *NIPS 7* (G. Tesauro, D. Touretzky, and T. Leen, eds.), Morgan Kaufmann.

H. Schwenk and M. Milgram (1994), "Structured diabolo-networks for handwritten character recognition," in *International Conference on Artificial Neural Networks*, pp. 985–988, Springer-Verlag.

P. Simard, B. Victorri, Y. LeCun, and J. Denker (1992), "Tangent prop - a formalism for specifying selected invariances in an adaptive network," in *NIPS 4* (J. Moody, S. Hanson, and R. Lippmann, eds.), pp. 895–903, Morgan Kaufmann.

P. Simard, Y. LeCun, and J. Denker (1993), "Efficient pattern recognition using a new transformation distance," in *NIPS 5* (S. Hanson, J. Cowan, and C. Giles, eds.), pp. 50–58, Morgan Kaufmann.

P. Simard (1994), "Efficient Computation of complex distance measures using hierarchical filtering," in *NIPS 6* (J.D. Cowan, G. Tesauro, and J. Alspector, eds.), pp. 50–58, Morgan Kaufmann.

A. Sperdutti and D.G. Stork (1995), "A rapid graph-based method for arbitrary transformation invariant pattern classification," in *NIPS 7* (G. Tesauro, D. Touretzky, and T. Leen, eds.), Morgan Kaufmann.